# Efficient Nonlinear Control with Actor-Tutor Architecture

**Kenji Doya***

ATR Human Information Processing Research Laboratories
2-2 Hikaridai, Seika-cho, Soraku-gun, Kyoto 619-02, Japan.

## Abstract

A new reinforcement learning architecture for nonlinear control is proposed. A direct feedback controller, or the actor, is trained by a value-gradient based controller, or the tutor. This architecture enables both efficient use of the value function and simple computation for real-time implementation. Good performance was verified in multi-dimensional nonlinear control tasks using Gaussian softmax networks.

## 1 INTRODUCTION

In the study of temporal difference (TD) learning in continuous time and space (Doya, 1996b), an optimal nonlinear feedback control law was derived using the gradient of the value function and the local linear model of the system dynamics. It was demonstrated in the simulation of a pendulum swing-up task that the value-gradient based control scheme requires much less learning trials than the conventional "actor-critic" control scheme (Barto et al., 1983).

In the actor-critic scheme, the actor, a direct feedback controller, improves its control policy stochastically using the TD error as the effective reinforcement (Figure 1a). Despite its relatively slow learning, the actor-critic architecture has the virtue of simple computation in generating control command. In order to train a direct controller while making efficient use of the value function, we propose a new reinforcement learning scheme which we call the "actor-tutor" architecture (Figure 1b).

*Current address: Kawato Dynamic Brain Project, JSTC. 2-2 Hikaridai, Seika-cho, Soraku-gun, Kyoto 619-02, Japan. E-mail: doya@erato.atr.co.jp

In the actor-tutor scheme, the optimal control command based on the current estimate of the value function is used as the target output of the actor. With the use of supervised learning algorithms (e.g., LMSE), learning of the actor is expected to be faster than in the actor-critic scheme, which uses stochastic search algorithms (e.g., $A_{RP}$). The simulation result below confirms this prediction. This hybrid control architecture provides a model of functional integration of motor-related brain areas, especially the basal ganglia and the cerebellum (Doya, 1996a).

## 2 CONTINUOUS TD LEARNING

First, we summarize the theory of TD learning in continuous time and space (Doya, 1996b), which is basic to the derivation of the proposed control scheme.

### 2.1 CONTINUOUS TD ERROR

Let us consider a continuous-time, continuous-state dynamical system

$$\frac{d\mathbf{x}(t)}{dt} = f(\mathbf{x}(t), \mathbf{u}(t)) \tag{1}$$

where $\mathbf{x} \in X \subset \mathbf{R}^n$ is the state and $\mathbf{u} \in U \subset \mathbf{R}^m$ is the control input (or the action). The reinforcement is given as the function of the state and the control

$$r(t) = r(\mathbf{x}(t), \mathbf{u}(t)). \tag{2}$$

For a given control law (or a policy)

$$\mathbf{u}(t) = \mu(\mathbf{x}(t)), \tag{3}$$

we define the "value function" of the state $\mathbf{x}(t)$ as

$$V^\mu(\mathbf{x}(t)) = \int_t^\infty \frac{1}{\tau} e^{-\frac{s-t}{\tau}} r(\mathbf{x}(s), \mathbf{u}(s)) ds, \tag{4}$$

where $\mathbf{x}(s)$ and $\mathbf{u}(s)$ ($t \le s < \infty$) follow the system dynamics (1) and the control law (3). Our goal is to find an optimal control law $\mu^*$ that maximizes $V^\mu(\mathbf{x})$ for any state $\mathbf{x} \in X$. Note that $\tau$ is the time constant of imminence-weighting, which is related to the discount factor $\gamma$ of the discrete-time TD as $\gamma = 1 - \frac{\Delta t}{\tau}$.

By differentiating (4) by $t$, we have a local consistency condition for the value function

$$\tau \frac{d}{dt} V^\mu(\mathbf{x}(t)) = V^\mu(\mathbf{x}(t)) - r(t). \tag{5}$$

Let $P(\mathbf{x}(t))$ be the prediction of the value function $V^\mu(\mathbf{x}(t))$ from $\mathbf{x}(t)$ by a neural network, or some function approximator that has enough capability of generalization. The prediction should be adjusted to minimize the inconsistency

$$\hat{r}(t) = r(t) - P(\mathbf{x}(t)) + \tau \frac{dP(\mathbf{x}(t))}{dt}, \tag{6}$$

which is a continuous version of the TD error. Because the boundary condition for the value function is given on the attractor set of the state space, correction of $P(\mathbf{x}(t))$ should be made backward into time. The correspondence between continuous-time TD algorithms and discrete-time TD($\lambda$) algorithms (Sutton, 1988) is shown in (Doya, 1996b).

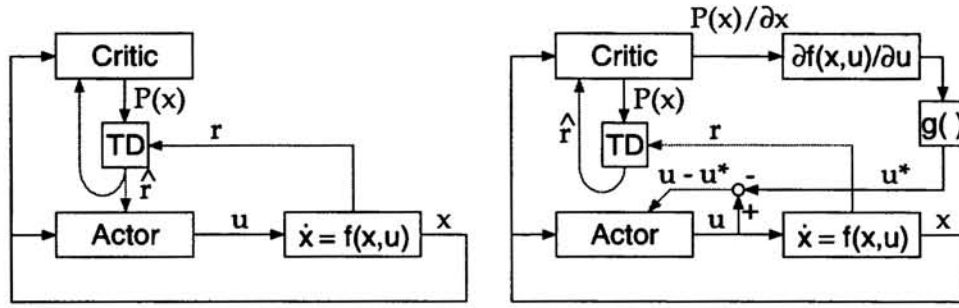

Figure 1:      (a) Actor-critic                          (b) Actor-tutor

## 2.2   OPTIMAL CONTROL BY VALUE GRADIENT

According to the principle of dynamic programming (Bryson and Ho, 1975), the local constraint for the value function $V^*$ for the optimal control law $\mu^*$ is given by the Hamilton-Jacobi-Bellman equation

$$V^*(t) = \max_{\mathbf{u}(t)\in U} \left[ r(\mathbf{x}(t), \mathbf{u}(t)) + \tau \frac{\partial V^*(\mathbf{x}(t))}{\partial \mathbf{x}} f(\mathbf{x}(t), \mathbf{u}(t)) \right]. \tag{7}$$

The optimal control $\mu^*$ is given by solving the maximization problem in the HJB equation, i.e.,

$$\frac{\partial r(\mathbf{x}, \mathbf{u})}{\partial \mathbf{u}} + \tau \frac{\partial V^*(\mathbf{x})}{\partial \mathbf{x}} \frac{\partial f(\mathbf{x}, \mathbf{u})}{\partial \mathbf{u}} = 0. \tag{8}$$

When the cost for each control variable is given by a convex potential function $G_j()$

$$r(\mathbf{x}, \mathbf{u}) = R(\mathbf{x}) - \sum_j G_j(u_j), \tag{9}$$

equation (8) can be solved using a monotonic function $g_j(x) = (G_j')^{-1}(x)$ as

$$u_j = g_j \left( \tau \frac{\partial V^*(\mathbf{x})}{\partial \mathbf{x}} \frac{\partial f(\mathbf{x}, \mathbf{u})}{\partial u_j} \right). \tag{10}$$

If the system is linear with respect to the input, which is the case with many mechanical systems, $\partial f(\mathbf{x}, \mathbf{u})/\partial u_j$ is independent of $\mathbf{u}$ and the above equation gives a closed-form optimal feedback control law $\mathbf{u} = \mu^*(\mathbf{x})$.

In practice, the optimal value function is unknown and we replace $V^*(\mathbf{x})$ with the current estimate of the value function $P(\mathbf{x})$

$$\mathbf{u} = g \left( \tau \frac{\partial P(\mathbf{x})}{\partial \mathbf{x}} \frac{\partial f(\mathbf{x}, \mathbf{u})}{\partial \mathbf{u}} \right). \tag{11}$$

While the system evolves with the above control law, the value function $P(\mathbf{x})$ is updated to minimize the TD error (6). In (11), the vector $\partial P(\mathbf{x})/\partial \mathbf{x}$ represents the desired motion direction in the state space and the matrix $\partial f(\mathbf{x}, \mathbf{u})/\partial \mathbf{u}$ transforms it into the action space. The function $g$, which is specified by the control cost, determines the amplitude of control output. For example, if the control cost $G$ is quadratic, then (11) reduces to a linear feedback control. A practically important case is when $g$ is a sigmoid, because this gives a feedback control law for a system with limited control amplitude, as in the examples below.

# 3 ACTOR-TUTOR ARCHITECTURE

It was shown in a task of a pendulum swing-up with limited torque (Doya, 1996b) that the above value-gradient based control scheme (11 can learn the task in much less trials than the actor-critic scheme. However, computation of the feedback command by (11) requires an on-line calculation of the gradient of the value function $\partial P(\mathbf{x})/\partial \mathbf{x}$ and its multiplication with the local linear model of the system dynamics $\partial f(\mathbf{x}, \mathbf{u})/\partial \mathbf{u}$, which can be too demanding for real-time implementation.

One solution to this problem is to use a simple direct controller network, as in the case of the actor-critic architecture. The training of the direct controller, or the actor, can be performed by supervised learning instead of trial-and-error learning because the target output of the controller is explicitly given by (11). Although computation of the target output may involve a processing time that is not acceptable for immediate feedback control, it is still possible to use its output for training the direct controller provided that there is some mechanism of short-term memory (e.g., eligibility trace in the connection weights).

Figure 1(b) is a schematic diagram of this "actor-tutor" architecture. The critic monitors the performance of the actor and estimates the value function. The "tutor" is a cascade of the critic, its gradient estimator, the local linear model of the system, and the differential model of control cost. The actor is trained to minimize the difference between its output and the tutor's output.

# 4 SIMULATION

We tested the performance of the actor-tutor architecture in two nonlinear control tasks; a pendulum swing-up task (Doya, 1996b) and the global version of a cart-pole balancing task (Barto et al., 1983).

The network architecture we used for both the actor and the critic was a Gaussian soft-max network. The output of the network is given by

$$y = \sum_{k=1}^{K} w_k b_k(\mathbf{x}),$$

$$b_k(\mathbf{x}) = \frac{\exp[-\sum_{i=1}^{n}(\frac{x_i - c_{ki}}{s_{ki}})^2]}{\sum_{l=1}^{K} \exp[-\sum_{i=1}^{n}(\frac{x_i - c_{li}}{s_{li}})^2]},$$

where $(c_{k1}, ..., c_{kn})$ and $(s_{k1}, ..., s_{kn})$ are the center and the size of the $k$-th basis function. It is in general possible to adjust the centers and sizes of the basis function, but in order to assure predictable transient behaviors, we fixed them in a grid. In this case, computation can be drastically reduced by factorizing the activation of basis functions in each input dimension.

## 4.1 PENDULUM SWING-UP TASK

The first task was to swing up a pendulum with a limited torque $|T| \leq T^{\max}$, which was about one fifth of the torque that was required to statically bring the pendulum up (Figure 2 (a)). This is a nonlinear control task in which the controller has to swing the pendulum several times at the bottom to build up enough momentum.

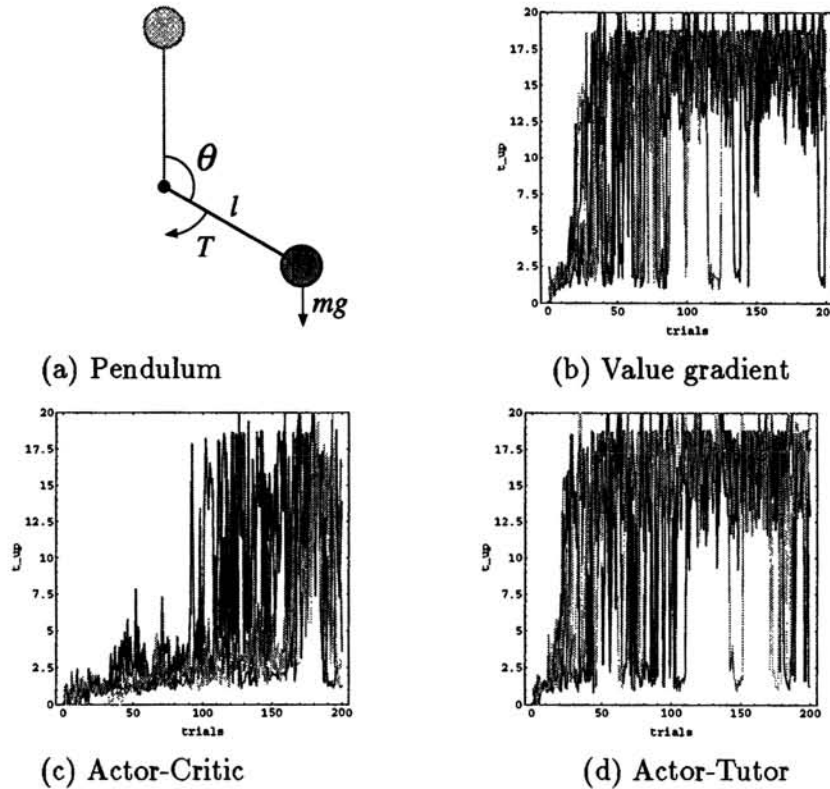

(a) Pendulum                          (b) Value gradient

(c) Actor-Critic                        (d) Actor-Tutor

Figure 2: Pendulum swing-up task. The dynamics of the pendulum (a) is given by $ml\ddot{\theta} = -\mu\dot{\theta} + mgl\sin\theta + T$. The parameters were $m = l = 1$, $g = 9.8$, $\mu = 0.01$, and $T^{\max} = 2.0$. The learning curves for value-gradient based optimal control (b), actor-critic (c), and actor-tutor (d); t_up is time during which $|\theta| < 45°$.

The state space for the pendulum $\mathbf{x} = (\theta, \omega)$ was 2D and we used $12 \times 12$ basis functions to cover the range $|\theta| \leq 180°$ and $|\omega| \leq 180°/s$. The reinforcement for the state was given by the height of the tip of the pendulum, i.e., $R(\mathbf{x}) = \cos\theta$ and the cost for control $G$ and the corresponding output sigmoid function $g$ were selected to match the maximal output torque $T^{\max}$.

Figures 2 (b), (c), and (d) show the learning curves for the value-gradient based control (11), actor critic, and actor-tutor control schemes, respectively. As we expected, the learning of the actor-tutor was much faster than that of the actor-critic and was comparable to the value-gradient based optimal control schemes.

## 4.2  CART-POLE SWING-UP TASK

Next we tested the learning scheme in a higher-dimensional nonlinear control task, namely, a cart-pole swing-up task (Figure 3). In the pioneering work of , the actor-critic system successfully learned the task of balancing the pole within $\pm 12°$ of the upright position while avoiding collision with the end of the cart track. The task we chose was to swing up the pole from an arbitrary angle and to balance it upright. The physical parameters of the cart-pole were the same as in (Barto et al., 1983) except that the length of the track was doubled to provide enough room for swinging.

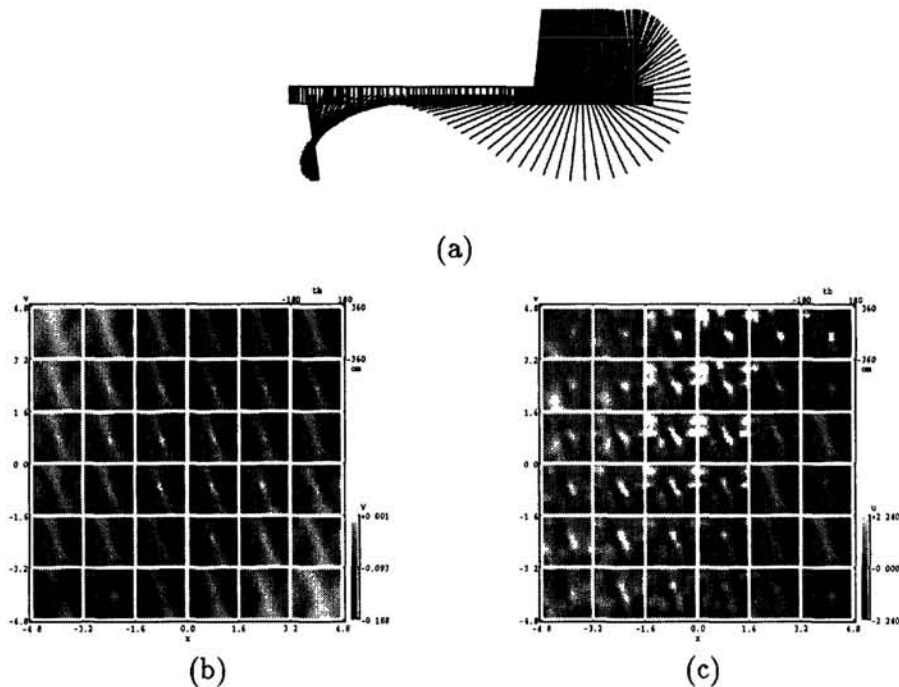

(a)

(b)                                        (c)

Figure 3: Cart-pole swing-up task. (a) An example of a swing-up trajectory. (b) Value function learned by the critic. (c) Feedback force learned by the actor. Each square in the plot shows a slice of the 4D state space parallel to the $(\theta, \omega)$ plane.

Figure 3 (a) shows an example of a successful swing up after 1500 learning trials with the actor-tutor architecture. We could not achieve a comparable performance with the actor-critic scheme within 3000 learning trials. Figures 3 (b) and (c) show the value function and the feedback force field, respectively, in the 4D state space $\mathbf{x} = (x, v, \theta, \omega)$, which were implemented in $6 \times 6 \times 12 \times 12$ Gaussian soft-max networks. We imposed symmetric constraints on both actor and critic networks to facilitate generalization. It can be seen that the paths to the upright position in the center of the track are represented as ridges in the value function.

## 5   DISCUSSION

The biggest problem in applying TD or DP to real-world control tasks is the curse of dimensionality, which makes both the computation for each data point and the numbers of data points necessary for training very high. The actor-tutor architecture provides a partial solution to the former problem in real-time implementation. The grid-based Gaussian soft-max basis function network was successfully used in a 4D state space. However, a more flexible algorithm that allocates basis functions only in the relevant parts of the state space may be necessary for dealing with higher-dimension systems (Schaal and Atkeson, 1996).

In the above simulations, we assumed that the local linear model of the system dynamics $\partial f(\mathbf{x}, \mathbf{u})/\partial \mathbf{u}$ was available. In preliminary experiments, it was verified that the critic, the system model, and the actor can be trained simultaneously.

The actor-tutor architecture resembles "feedback error learning" (Kawato et al., 1987) in the sense that a nonlinear controller is trained by the output of anther controller. However, the actor-tutor scheme can be applied to a highly nonlinear control task to which it is difficult to prepare a simple linear feedback controller.

Motivated by the performance of the actor-tutor architecture and the recent physiological and fMRI experiments on the brain activity during the course of motor learning (Hikosaka et al., 1996; Imamizu et al., 1996), we proposed a framework of functional integration of the basal ganglia, the cerebellum, and cerebral motor areas (Doya, 1996a). In this framework, the basal ganglia learns the value function $P(\mathbf{x})$ (Houk et al., 1994) and generates the desired motion direction based on its gradient $\partial P(\mathbf{x})/\partial \mathbf{x}$. This is transformed into a motor command by the "transpose model" of the motor system $(\partial f(\mathbf{x},\mathbf{u})/\partial \mathbf{u})^T$ in the lateral cerebellum (cerebrocerebellum). In early stages of learning, this output is used for control, albeit its feedback latency is long. As the subject repeats the same task, a direct controller is constructed in the medial and intermediate cerebellum (spinocerebellum) with the above motor command as the teacher. The direct controller enables quick, near-automatic performance with less cognitive load in other parts of the brain.

## References

Barto, A. G., Sutton, R. S., and Anderson, C. W. (1983). Neuronlike adaptive elements that can solve difficult learning control problems. *IEEE Transactions on Systems, Man, and Cybernetics*, 13:834–846.

Bryson, Jr., A. E. . and Ho, Y.-C. (1975). *Applied Optimal Control.* Hemisphere Publishing, New York, 2nd edition.

Doya, K. (1996a). An integrated model of basal ganglia and cerebellum in sequential control tasks. *Society for Neuroscience Abstracts*, 22:2029.

Doya, K. (1996b). Temporal difference learning in continuous time and space. In Touretzky, D. S., Mozer, M. C., and Hasselmo, M. E., editors, *Advances in Neural Information Processing Systems 8*, pages 1073–1079. MIT Press, Cambridge, MA.

Hikosaka, O., Miyachi, S., Miyashita, K., and Rand, M. K. (1996). Procedural learning in monkeys — Possible roles of the basal ganglia. In Ono, T., McNaughton, B. L., Molotchnikoff, S., Rolls, E. T., and Nishijo, H., editors, *Perception, Memory and Emotion: Frontiers in Neuroscience*, pages 403–420. Pergamon, Oxford.

Houk, J. C., Adams, J. L., and Barto, A. G. (1994). A model of how the basal ganglia generate and use neural signals that predict reinforcement. In Houk, J. C., Davis, J. L., and Beiser, D. G., editors, *Models of Information Processing in the Basal Ganglia*, pages 249–270. MIT Press, Cambrigde, MA.

Imamizu, H., Miyauchi, S., Sasaki, Y., Takino, R., Putz, B., and Kawato, M. (1996). A functional MRI study on internal models of dynamic transformations during learning a visuomotor task. *Society for Neuroscience Abstracts*, 22:898.

Kawato, M., Furukawa, K., and Suzuki, R. (1987). A hierarchical neural network model for control and learning of voluntary movement. *Biological Cybernetics*, 57:169–185.

Schaal, S. and Atkeson, C. C. (1996). From isolation to cooperation: An alternative view of a system of experts. In Touretzky, D. S., Mozer, M. C., and Hasselmo, M. E., editors, *Advances in Neural Information Processing Systems 8*, pages 605–611. MIT Press, Cambridge, MA, USA.

Sutton, R. S. (1988). Learning to predict by the methods of temporal difference. *Machine Learning*, 3:9–44.
